# Associative Decorrelation Dynamics: A Theory of Self-Organization and Optimization in Feedback Networks

**Dawei W. Dong***

Lawrence Berkeley Laboratory
University of California
Berkeley, CA 94720

## Abstract

This paper outlines a dynamic theory of development and adaptation in neural networks with feedback connections. Given input ensemble, the connections change in strength according to an associative learning rule and approach a stable state where the neuronal outputs are decorrelated. We apply this theory to primary visual cortex and examine the implications of the dynamical decorrelation of the activities of orientation selective cells by the intracortical connections. The theory gives a unified and quantitative explanation of the psychophysical experiments on orientation contrast and orientation adaptation. Using only one parameter, we achieve good agreements between the theoretical predictions and the experimental data.

## 1 Introduction

The mammalian visual system is very effective in detecting the orientations of lines and most neurons in primary visual cortex selectively respond to oriented lines and form orientation columns [1]. Why is the visual system organized as such? We

believe that the visual system is self-organized, in both long term development and short term adaptation, to ensure the optimal information processing.

Linsker applied Hebbian learning to model the development of orientation selectivity and later proposed a principle of maximum information preservation in early visual pathways [2]. The focus of his work has been on the feedforward connections and in his model the feedback connections are isotropic and unchanged during the development of orientation columns; but the actual circuitry of visual cortex involves extensive, columnar specified feedback connections which exist even before functional columns appear in cat striate cortex [3].

Our earlier research emphasized the important role of the feedback connections in the development of the columnar structure in visual cortex. We developed a theoretical framework to help understand the dynamics of Hebbian learning in feedback networks and showed how the columnar structure originates from symmetry breaking in the development of the feedback connections (intracortical, or lateral connections within visual cortex) [4].

Figure 1 illustrates our theoretical predictions. The intracortical connections break symmetry and develop strip-like patterns with a characteristic wave length which is comparable to the developed intracortical inhibitory range and the LGN-cortex afferent range (left). The feedforward (LGN-cortex) connections develop under the influence of the symmetry breaking development of the intracortical connections. The developed feedforward connections for each cell form a receptive field which is orientation selective and nearby cells have similar orientation preference (right). Their orientations change in about the same period as the strip-like pattern of the intracortical connections.

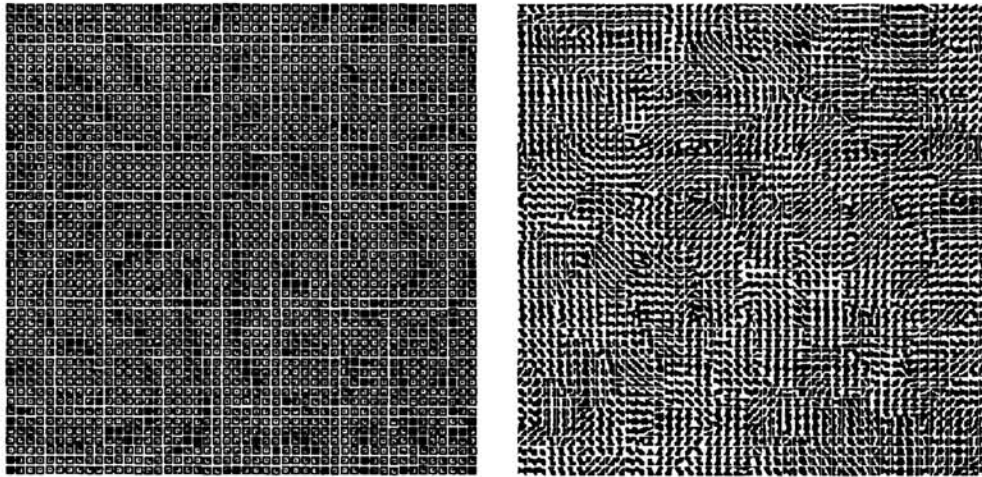

Figure 1: The results of the development of visual cortex with feedback connections. The simulated cortex consists of 48 × 48 neurons, each of which connects to 5 × 5 other cortical neurons (left) and receives inputs from 7 × 7 LGN neurons (right). In this figure, white indicates positive connections and black indicates negative connections. One can see that the change of receptive field's orientation (right) is highly correlated with the strip-like pattern of intracortical connections (left).

Many aspects of our theoretical predictions agree qualitatively with neurobiological observations in primary visual cortex. Another way to test the idea of optimal

information processing or any self-organization theory is through quantitative psychophysical studies. The idea is to look for changes in perception following changes in input environments. The psychophysical experiments on orientation illusions offer some opportunities to test our theory on orientation selectivity.

Orientation illusions are the effects that the perceived orientations of lines are affected by the neighboring (in time or space) oriented stimuli, which have been observed in many psychophysical experiments and were attributed to the inhibitory interactions between channels tuned to different orientations [5]. But there is no unified and quantitative explanation. Neurophysiological evidences support our earlier computational model in which intracortical inhibition plays the role of gain-control in orientation selectivity [6]. But in order for the gain-control mechanism to be effective to signals of different statistics, the system has to develop and adapt in different environments.

In this paper we examine the implication of the hypothesis that the intracortical connections dynamically decorrelate the activities of orientation selective cells, i.e., the intracortical connections are actively adapted to the visual environment, such that the output activities of orientation selective cells are decorrelated. The dynamics which ensures such decorrelation through associative learning is outlined in the next section as the theoretical framework for the development and the adaptation of intracortical connections. We only emphasize the feedback connections in the following sections and assume that the feedforward connections developed orientation selectivities based on our earlier works. The quantitative comparisons of the theory and the experiments are presented in section 3.

## 2  Associative Decorrelation Dynamics

There are two different kinds of variables in neural networks. One class of variables represents the activity of the nerve cells, or neurons. The other class of variables describes the synapses, or connections, between the nerve cells. A complete model of an adaptive neural system requires two sets of dynamical equations, one for each class of variables, to specify the evolution and behavior of the neural system.

The set of equations describing the change of the state of activity of the neurons is

$$a\frac{dV_i}{dt} = -V_i + \sum_j T_{ij} V_j + I_i \tag{1}$$

in which $a$ is a time constant, $T_{ij}$ is the strength of the synaptic connection from neuron $j$ to neuron $i$, and $I_i$ is the additional feedforward input to the neuron besides those described by the feedback connection matrix $T_{ij}$. A second set of equations describes the way the synapses change with time due to neuronal activity. The learning rule proposed here is

$$B\frac{dT_{ij}}{dt} = (V_i - V_i')I_j \tag{2}$$

in which $B$ is a time constant and $V_i'$ is the feedback learning signal as described in the following.

The feedback learning signal $V_i'$ is generated by a Hopfield type associative memory network: $V_i' = \sum_j T_{ij}' V_j$, in which $T_{ij}'$ is the strength of the associative connection

from neuron $j$ to neuron $i$, which is the recent correlation between the neuronal activities $V_i$ and $V_j$ determined by Hebbian learning with a decay term [4]

$$B' \frac{dT'_{ij}}{dt} = -T'_{ij} + V_i V_j \qquad (3)$$

in which $B'$ is a time constant. The $V'_i$ and $T'_{ij}$ are only involved in learning and do not directly affect the network outputs.

It is straight forward to show that when the time constants $B >> B' >> a$, the dynamics reduces to

$$B \frac{d\mathbf{T}}{dt} = (\mathbf{1} - <\mathbf{V}\mathbf{V}^T>) <\mathbf{V}\mathbf{I}^T> \qquad (4)$$

where bold-faced quantities are matrices and vectors and $<>$ denotes ensemble average. It is not difficult to show that this equation has a Lyapunov or "energy" function

$$L = \mathrm{Tr}(\mathbf{1} - <\mathbf{V}\mathbf{V}^T>)(\mathbf{1} - <\mathbf{V}\mathbf{V}^T>)^T \qquad (5)$$

which is lower bounded and satisfies

$$\frac{dL}{dt} \le 0 \quad \text{and} \quad \frac{dL}{dt} = 0 \quad \rightarrow \quad \frac{dT_{ij}}{dt} = 0 \quad \text{for all } i, j \qquad (6)$$

Thus the dynamics is stable. When it is stable, the output activities are decorrelated,

$$<\mathbf{V}\mathbf{V}^T> = \mathbf{1} \qquad (7)$$

The above equation shows that this dynamics always leads to a stable state where the neuronal activities are decorrelated and their correlation matrix is orthonormal. Yet the connections change in an associative fashion — equation (2) and (3) are almost Hebbian. That is why we call it associative decorrelation dynamics. From information processing point of view, a network, self-organized to satisfy equation (7), is optimized for Gaussian input ensembles and white output noises [7].

**Linear First Order Analysis**

In applying our theory of associative decorrelation dynamics to visual cortex to compare with the psychophysical experiments on orientation illusions, the linear first-order approximation is used, which is

$$\begin{aligned} \mathbf{T} &= \mathbf{T}^0 + \delta\mathbf{T}, \quad \mathbf{T}^0 = \mathbf{0}, \quad \delta\mathbf{T} \propto - <\mathbf{I}\,\mathbf{I}^T> \\ \mathbf{V} &= \mathbf{V}^0 + \delta\mathbf{V}, \quad \mathbf{V}^0 = \mathbf{I}, \quad \delta\mathbf{V} = \mathbf{T}\mathbf{I} \end{aligned} \qquad (8)$$

where it is assumed that the input correlations are small. It is interesting to notice that the linear first-order approximation leads to anti-Hebbian feedback connections: $T_{ij} \propto - <I_i I_j>$ which is guaranteed to be stable around $\mathbf{T} = \mathbf{0}$ [8].

## 3   Quantitative Predictions of Orientation Illusions

The basic phenomena of orientation illusions are demonstrated in figure 2 (left). On the top, is the effect of orientation contrast (also called tilt illusion): within the two surrounding circles there are tilted lines; the orientation of a center rectangle

appears rotated to the opposite side of its surrounding tilt. Both the two rectangles and the one without surround (at the left-center of this figure) are, in fact, exactly same. On the bottom, is the effect of orientation adaptation (also called tilt aftereffect): if one fixates at the small circle in one of the two big circles with tilted lines for 20 seconds or so and then look at the rectangle without surround, the orientation of the lines of the rectangle appears tilted to the opposite side.

These two effects of orientation illusions are both in the direction of repulsion: the apparent orientation of a line is changed to increase its difference from the inducing line. Careful experimental measurements also revealed that the angle with the inducing line is $\sim 10°$ for maximum orientation adaptation effect [9] but $\sim 20°$ for orientation contrast [10].

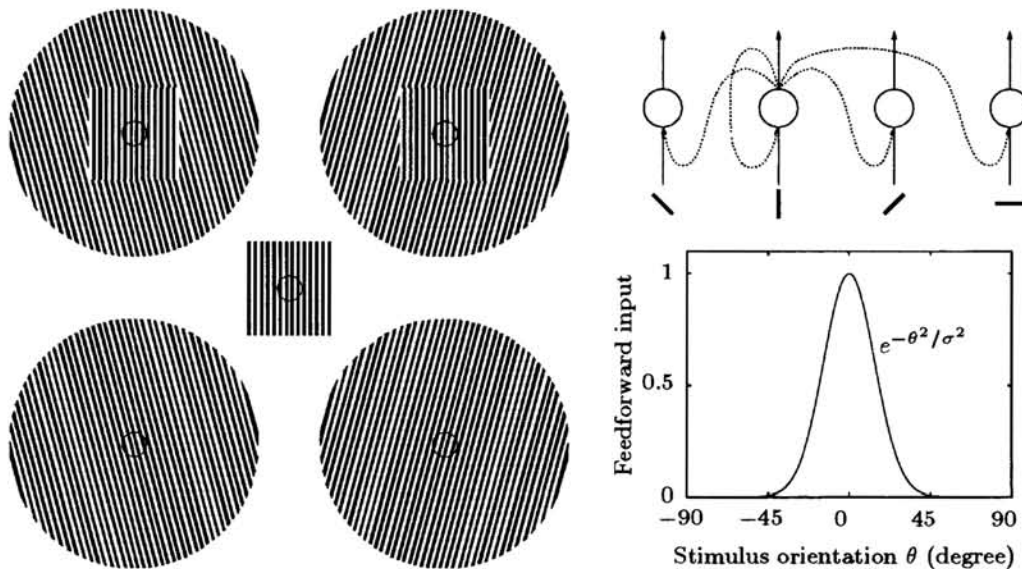

Figure 2: The effects of orientation contrast (upper-left) and orientation adaptation (lower-left) are attributed to feedback connections between cells tuned to different orientations (upper-right, network; lower-right, tuning curve).

Orientation illusions are attributed to the feedback connections between orientation selective cells. This is illustrated in figure 2 (right). On the top is the network of orientation selective cells with feedback connections. Only four cells are shown. From the left, they receive orientation selective feedforward inputs optimal at $-45°$, $0°, 45°$, and $90°$, respectively. The dotted lines represent the feedback connections (only the connections from the second cell are drawn). On the bottom is the orientation tuning curve of the feedforward input for the second cell, optimally tuned to stimulus of $0°$ (vertical), which is assumed to be Gaussian of width $\sigma = 20°$. Because of the feedback connections, the output of the second cell will have different tuning curves from its feedforward input, depending on the activities of other cells.

For primary visual cortex, we suppose that there are orientation selective neurons tuned to all orientations. It is more convenient to use the continuous variable $\theta$ instead of the index $i$ to represent neuron which is optimally tuned to the orientation of angle $\theta$. The neuronal activity is represented by $V(\theta)$ and the feedforward input to each neuron is represented by $I(\theta)$. The feedforward input itself is orientation

selective: given a visual stimulus of orientation $\theta_0$, the input is

$$I(\theta) = e^{-(\theta-\theta_0)^2/\sigma^2} \tag{9}$$

This kind of the orientation tuning has been measured by experiments (for references, see [6]). Various experiments give a reasonable tuning width around $20^o$ ($\sigma = 20^o$ is used for all the predictions).

**Predicted Orientation Adaptation**

For the orientation adaptation to stimulus of angle $\theta_0$, substituting equation (9) into equation (8), it is not difficult to derive that the network response to stimulus of angle 0 (vertical) is changed to

$$V(\theta) = e^{-\theta^2/\sigma^2} - \alpha e^{-(\theta-\theta_0)^2/\sigma^2} e^{-\theta_0^2/2\sigma^2} \tag{10}$$

in which $\sigma$ is the feedforward tuning width chosen to be $20^o$ and $\alpha$ is the parameter of the strength of decorrelation feedback.

The theoretical curve of perceived orientation $\phi(\theta_0)$ is derived by assuming the maximum likelihood of the the neural population, i.e., the perceived angle $\phi$ is the angle at which $V(\theta)$ is maximized. It is shown in figure 3 (right). The solid line is the theoretical curve and the experimental data come from [9] (they did not give the errors, the error bars are of our estimation $\sim 0.2^o$). The parameter obtained through $\chi^2$ fit is the strength of decorrelation feedback: $\alpha = 0.42$.

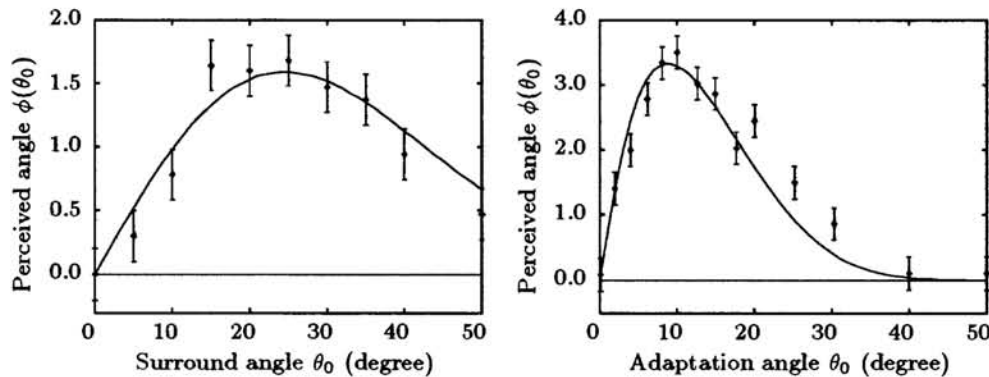

Figure 3: Quantitative comparison of the theoretical predictions with the experimental data of orientation contrast (left) and orientation adaptation (right).

It is very interesting that we can derive a relationship which is independent of the parameter of the strength of decorrelation feedback $\alpha$,

$$(\theta_0 - \phi_m)(3\theta_0 - 2\phi_m) = \sigma^2 \tag{11}$$

in which $\theta_0$ is the adaptation angle at which the tilt aftereffect is most significant and $\phi_m$ is the perceived angle.

**Predicted Orientation Contrast**

For orientation contrast, there is no specific adaptation angle, i.e., the network has developed in an environment of all possible angles. In this case, when the surround is of angle $\theta_0$, the network response to a stimulus of angle $\theta_1$ is

$$V(\theta) = e^{-(\theta-\theta_1)^2/\sigma^2} - \alpha e^{-(\theta-\theta_0)^2/3\sigma^2} \tag{12}$$

in which $\sigma$ and $\alpha$ has the same meaning as for orientation adaptation. Again assuming the maximum likelihood, $\phi(\theta_0)$, the stimulus angle $\theta_1$ at which it is perceived as angle 0, is derived and shown in figure 3 (left). The solid line is the theoretical curve and the experimental data come from [10] and their estimated error is $\sim 0.2^o$. The parameter obtained through $\chi^2$ fit is the strength of decorrelation feedback: $\alpha = 0.32$.

We can derive the peak position $\theta_0$, i.e., the surrounding angle $\theta_0$ at which the orientation contrast is most significant,

$$\frac{2}{3}\theta_0^2 = \sigma^2 \tag{13}$$

For $\sigma = 20^o$, one immediately gets $\theta_0 = 24^o$. This is in good agreement with experiments, most people experience the maximum effect of orientation contrast around this angle.

Our theory predicts that the peak position of the surround angle for orientation contrast should be constant since the orientation tuning width $\sigma$ is roughly the same for different human observers and is not going to change much for different experimental setups. But the peak value of the perceived angle is not constant since the decorrelation feedback parameter $\alpha$ is not necessarily same, indeed, it could be quite different for different human observers and different experimental setups.

## 4   Discussion

First, we want to emphasis that in all the comparisons, the same tuning width $\sigma$ is used and the strength of decorrelation feedback $\alpha$ is the only fit parameter. It does not take much imagination to see that the quantitative agreements between the theory and the experiments are good. Further more, we derived the relationships for the maximum effects, which are independent of the parameter $\alpha$ and have been partially confirmed by the experiments.

Recent neurophysiological experiments revealed that the surrounding lines did influence the orientation selectivity of cells in primary visual cortex of the cat [11]. Those single cell experiments land further support to our theory. But one should be cautioned that the cells in our theory should be considered as the average over a large population of cells in cortex.

The theory not only explains the first order effects which are dominant in angle range of $0^o$ to $50^o$, as shown here, but also accounts for the second order effects which can be seen in $50^o$ to $90^o$ range, where the sign of the effects is reversed. The theory also makes some predictions for which not much experiment has been done yet, for example, the prediction about how orientation contrast depends on the distance of surrounding stimuli from the test stimulus [7].

Finally, this is not merely a theory for the development and the adaptation of orientation selective cells, it can account for effect such as human vision adaptation to colors as well [7]. We can derive the same equation as Atick *etal* [12] which agrees with the experiment on the appearance of color hue after adaptation. We believe that future psychophysical experiments could give us more quantitative results to further test our theory and help our understanding of neural systems in general.

## Acknowledgements

This work was supported in part by the Director, Office of Energy Research, Division of Nuclear Physics of the Office of High Energy and Nuclear Physics of the U.S. Department of Energy under Contract No. DE-AC03-76SF00098.

## Footnotes

*Present address: Rockefeller University, B272, 1230 York Avenue, NY, NY 10021-6399.

## References

[1] Hubel DH, Wiesel TN, 1962 Receptive fields, binocular interactions, and functional architecture in the cat's visual cortex *J Physiol (London)* **160**, 106–54. — 1963 Shape and arrangement of columns in cat's striate cortex *J Physiol (London)* **165**, 559–68.

[2] Linsker R, 1986 From basic network principles to neural architecture ... *Proc Natl Acad Sci USA* **83**, 7508 8390 8779. —, 1989 An application of the principle of maximum information preservation to linear systems *Advances in Neural Information Processing Systems 1, Touretzky DS, ed, Morgan Kaufman, San Mateo, CA* 186–94.

[3] Gilbert C, Wiesel T, 1989 Columnar Specificity of intrinsic horizontal and corticocortical connections in cat visual cortex *J Neurosci* **9(7)**, 2432–42. Luhmann HJ, Martinez L, Singer W, 1986 Development of horizontal intrinsic connections in cat striate cortex *Exp Brain Res* **63**, 443–8.

[4] Dong DW, 1991 Dynamic properties of neural network with adapting synapses *Proc International Joint Conference on Neural Networks, Seattle,* **2**, 255–260. —, 1991 Dynamic Properties of Neural Networks *Ph D thesis, University Microfilms International, Ann Arbor, MI*. Dong DW, Hopfield JJ, 1992 Dynamic properties of neural networks with adapting synapses *Network: Computation in Neural Systems,* **3(3)**, 267–83.

[5] Gibson JJ, Radner M, 1937 Adaptation, after-effect and contrast in the perception of tilted lines *J of Exp Psy* **20**, 453–67. Carpenter RHS, Blakemore C, 1973 Interactions between orientations in human vision *Exp Brain Res* **18**, 287–303. Tolhurst DJ, Thompson PG, 1975 Orientation illusions and after-effects: Inhibition between channels *Vis Res* **15**, 967–72. Barlow HB, Foldiak P, 1989 Adaptation and decorrelation in the cortex *The Computing Neuron, Durbin R, Miall C, Mitchison G, eds, Addison-Wesley, New York, NY*.

[6] Wehmeier U, Dong DW, Koch C, Van Essen DC, 1989 Modeling the mammalian visual system *Methods in Neuronal Modeling: From Synapses to Networks, Koch C, Segev I, eds, MIT Press, Cambridge, MA* 335–60.

[7] Dong DW, 1993 Associative Decorrelation Dynamics in Visual Cortex *Lawrence Berkeley Laboratory Technical Report* LBL-34491.

[8] Dong DW, 1993 Anti-Hebbian dynamics and total recall of associative memory *Proc World Congress on Neural Networks, Portland,* **2**, 275–9.

[9] Campbell FW, Maffei L, 1971 The tilt after-effect: a fresh look *Vis Res* **11**, 833–40.

[10] Westheimer G, 1990 Simultaneous orientation contrast for lines in the human fovea *Vis Res* **30**, 1913–21.

[11] Gilbert CD, Wiesel TN, 1990 The influence of contextual stimuli on the orientation selectivity of cells in primary visual cortex of the cat *Vis Res* **30**, 1689–701.

[12] Atick JJ, Li Z, Redlich AN, 1993 What does post-adaptation color appearance reveal about cortical color representation *Vis Res* **33**, 123–9.